# Individualized ROI Optimization via Maximization of Group-wise Consistency of Structural and Functional Profiles

[1, 2*]Kaiming Li, [1]Lei Guo, [3]Carlos Faraco, [2]Dajiang Zhu, [2]Fan Deng, [1]Tuo Zhang, [1]Xi Jiang, [1]Degang Zhang, [1]Hanbo Chen, [1]Xintao Hu, [3]Steve Miller, [2]Tianming Liu

[1]School of Automation, Northwestern Polytechnical University,China;[2]Department of Computer Science, the University of Georgia, USA;[3]Department of Psychology, the University of Georgia, USA; *Email:*likaiming@gmail.com*

## Abstract

Functional segregation and integration are fundamental characteristics of the human brain. Studying the connectivity among segregated regions and the dynamics of integrated brain networks has drawn increasing interest. A very controversial, yet fundamental issue in these studies is how to determine the best functional brain regions or ROIs (regions of interests) for individuals. Essentially, the computed connectivity patterns and dynamics of brain networks are very sensitive to the locations, sizes, and shapes of the ROIs. This paper presents a novel methodology to optimize the locations of an individual's ROIs in the working memory system. Our strategy is to formulate the individual ROI optimization as a group variance minimization problem, in which group-wise functional and structural connectivity patterns, and anatomic profiles are defined as optimization constraints. The optimization problem is solved via the simulated annealing approach. Our experimental results show that the optimized ROIs have significantly improved consistency in structural and functional profiles across subjects, and have more reasonable localizations and more consistent morphological and anatomic profiles.

## 1    Introduction

The human brain's function is segregated into distinct regions and integrated via axonal fibers [1]. Studying the connectivity among these regions and modeling their dynamics and interactions has drawn increasing interest and effort from the brain imaging and neuroscience communities [2-6]. For example, recently, the Human Connectome Project [7] and the 1000 Functional Connectomes Project [8] have embarked to elucidate large-scale connectivity patterns in the human brain. For traditional connectivity analysis, a variety of models including DCM (dynamics causal modeling), GCM (Granger causality modeling) and MVA (multivariate autoregressive modeling) are proposed [6, 9-10] to model the interactions of the ROIs. A fundamental issue in these studies is how to accurately identify the ROIs, which are the structural substrates for measuring connectivity. Currently, this is still an open, urgent, yet challenging problem in many brain imaging applications. From our perspective, the major challenges come from uncertainties in ROI boundary definition, the tremendous variability across individuals, and high nonlinearities within and around ROIs.

Current approaches for identifying brain ROIs can be broadly classified into four categories. The first is manual labeling by experts using their domain knowledge. The second is a data-driven clustering of ROIs from the brain image itself. For instance, the ReHo (regional homogeneity) algorithm [11] has been used to identify regional homogeneous regions as ROIs. The third is to predefine ROIs in a template brain, and warp them back to the individual space using image registration [12]. Lastly, ROIs can be defined from the activated regions observed during a task-based fMRI paradigm. While fruitful results have been achieved using these approaches, there are various limitations. For instance, manual labeling is difficult to implement for large datasets and may be vulnerable to inter-subject and intra-subject variation;

it is difficult to build correspondence across subjects using data-driven clustering methods; warping template ROIs back to individual space is subject to the accuracy of warping techniques and the anatomical variability across subjects.

Even identifying ROIs using task-based fMRI paradigms, which is regarded as the standard approach for ROI identification, is still an open question. It was reported in [13] that many imaging-related variables including scanner vender, RF coil characteristics (phase array vs. volume coil), k-space acquisition trajectory, reconstruction algorithms, susceptibility-induced signal dropout, as well as field strength differences, contribute to variations in ROI identification. Other researchers reported that spatial smoothing, a common preprocessing technique in fMRI analysis to enhance SNR, may introduce artificial localization shifts (up to 12.1mm for Gaussian kernel volumetric smoothing) [14] or generate overly smoothed activation maps that may obscure important details [15]. For example, as shown in Fig.1a, the local maximum of the ROI was shifted by 4mm due to the spatial smoothing process. Additionally, its structural profile (Fig.1b) was significantly altered. Furthermore, group-based activation maps may show different patterns from an individual's activation map; Fig.1c depicts such differences. The top panel is the group activation map from a working memory study, while the bottom panel is the activation map of one subject in the study. As we can see from the highlighted boxes, the subject has less activated regions than the group analysis result. In conclusion, standard analysis of task-based fMRI paradigm data is inadequate to accurately localize ROIs for each individual.

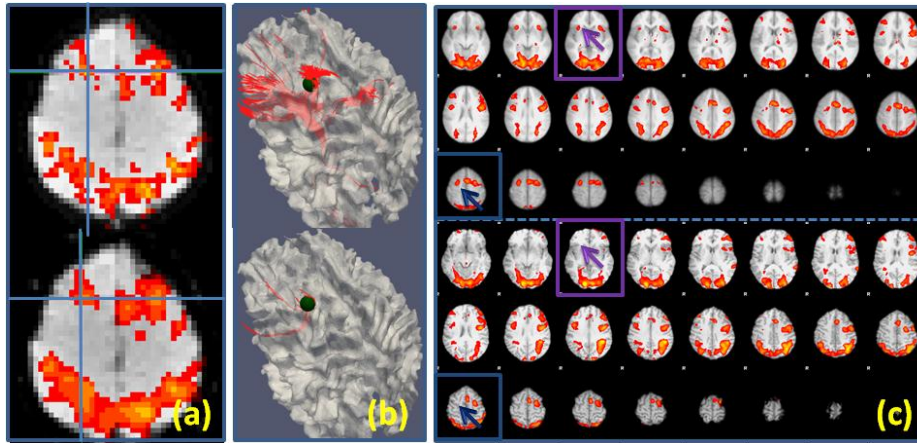

Fig.1. (a): Local activation map maxima (marked by the cross) shift of one ROI due to spatial volumetric smoothing. The top one was detected using unsmoothed data while the bottom one used smoothed data (FWHM: 6.875mm). (b): The corresponding fibers for the ROIs in (a). The ROIs are presented using a sphere (radius: 5mm). (c): Activation map differences between the group (top) and one subject (bottom). The highlighted boxes show two of the missing activated ROIs found from the group analysis.

Without accurate and reliable individualized ROIs, the validity of brain connectivity analysis, and computational modeling of dynamics and interactions among brain networks, would be questionable. In response to this fundamental issue, this paper presents a novel computational methodology to optimize the locations of an individual's ROIs initialized from task-based fMRI. We use the ROIs identified in a block-based working memory paradigm as a test bed application to develop and evaluate our methodology. The optimization of ROI locations was formulated as an energy minimization problem, with the goal of jointly maximizing the group-wise consistency of functional and structural connectivity patterns and anatomic profiles. The optimization problem is solved via the well-established simulated annealing approach. Our experimental results show that the optimized ROIs achieved our optimization objectives and demonstrated promising results.

## 2    Materials and Methods

### 2.1    Data acquisition and preprocessing

Twenty-five university students were recruited to participate in this study. Each participant performed an fMRI modified version of the OSPAN task (3 block types: OSPAN, Arithmetic, and Baseline) while fMRI data was acquired. DTI scans were also acquired for each participant. FMRI and DTI scans were acquired on a 3T GE Signa scanner. Acquisition parameters were as follows : fMRI: 64x64 matrix, 4mm slice thickness, 220mm FOV, 30 slices, TR=1.5s, TE=25ms, ASSET=2; DTI: 128x128 matrix, 2mm slice thickness, 256mm FOV, 60 slices, TR=15100ms, TE= variable, ASSET=2, 3 B0 images, 30 optimized gradient directions, b-value=1000). Each participant's fMRI data was analyzed using FSL. Individual activation map reflecting the OSPAN (OSPAN > Baseline) contrast was used. In total, we identified the 16 highest activated ROIs, including left and right insula, left and right medial frontal gyrus, left and right precentral gyrus, left and right paracingulate gyrus, left and right

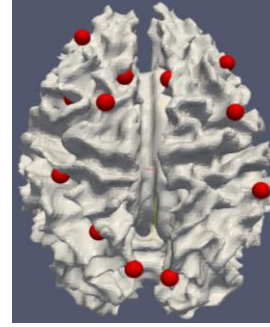

Fig.2. working memory ROIs mapped on a WM/GM surface

dorsolateral prefrontal cortex, left and right inferior parietal lobule, left occipital pole, right frontal pole, right lateral occipital gyrus, and left and right precuneus. Fig.2 shows the 16 ROIs mapped onto a WM(white matter)/GM(gray matter) cortical surface. For some individuals, there may be missing ROIs on their activation maps. Under such condition, we adapted the group activation map as a guide to find these ROIs using linear registration.

DTI pre-processing consisted of skull removal, motion correction, and eddy current correction. After the pre-processing, fiber tracking was performed using MEDINRIA (FA threshold: 0.2; minimum fiber length: 20). Fibers were extended along their tangent directions to reach into the gray matter when necessary. Brain tissue segmentation was conducted on DTI data by the method in [16] and the cortical surface was reconstructed from the tissue maps using the marching cubes algorithm. The cortical surface was parcellated into anatomical regions using the HAMMER tool [17]. DTI space was used as the standard space from which to generate the GM (gray matter) segmentation and from which to report the ROI locations on the cortical surface. Since the fMRI and DTI sequences are both EPI (echo planar imaging) sequences, their distortions tend to be similar and the misalignment between DTI and fMRI images is much less than that between T1 and fMRI images [18]. Co-registration between DTI and fMRI data was performed using FSL FLIRT [12]. The activated ROIs and tracked fibers were then mapped onto the cortical surface for joint modeling.

## 2.2    Joint modeling of anatomical, structural and functional profiles

Despite the high degree of variability across subjects, there are several aspects of regularity on which we base the proposed solution. Firstly, across subjects, the functional ROIs should have similar anatomical locations, e.g., similar locations in the atlas space. Secondly, these ROIs should have similar structural connectivity profiles across subjects. In other words, fibers penetrating the same functional ROIs should have at least similar target regions across subjects. Lastly, individual networks identified by task-based paradigms, like the working memory network we adapted as a test bed in this paper, should have similar functional connectivity pattern across subjects. The neuroscience bases of the above premises include: 1) structural and functional brain connectivity are closely related [19], and cortical gyrification and axongenesis processes are closely coupled [20]; Hence, it is reasonable to put these three types of information in a joint modeling framework. 2) Extensive studies have already demonstrated the existence of a common structural and functional architecture of the human brain [21, 22], and it makes sense to assume that the working memory network has similar structural and functional connectivity patterns across individuals.

Based on these premises, we proposed to optimize the locations of individual functional ROIs by jointly modeling anatomic profiles, structural connectivity patterns, and functional connectivity patterns, as illustrated in Fig 3. The

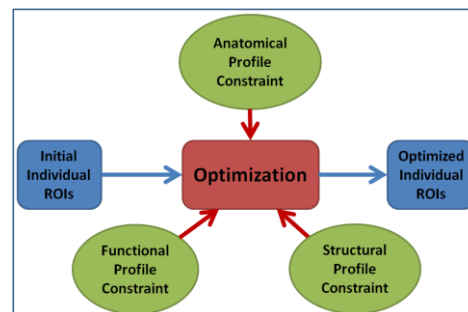

Fig.3. ROIs optimization scheme.

goal was to minimize the group-wise variance (or maximize group-wise consistency) of these jointly modeled profiles. Mathematically, we modeled the group-wise variance as energy $E$ as follows. A ROI from fMRI analysis was mapped onto the surface, and is represented by a center vertex and its neighborhood. Suppose $R_{ij}$ is the ROI region $j$ on the cortical surface of subject $i$ identified in Section 2.1; we find a corresponding surface ROI region $S_{ij}$ so that the energy $E$ (contains energy from $n$ subjects, each with $m$ ROIs) is minimized:

$$E = E_a \left( \lambda \frac{E_c - M_{E_c}}{\sigma_{E_c}} + (1 - \lambda) \frac{E_f - M_{E_f}}{\sigma_{E_f}} \right) \qquad (1)$$

where $E_a$ is the anatomical constraint; $E_c$ is the structural connectivity constraint, $M_{E_c}$ and $\sigma_{E_c}$ are the mean and standard deviation of $E_c$ in the searching space; $E_f$ is the functional connectivity constraint, $M_{E_f}$ and $\sigma_{E_f}$ are the mean and standard deviation of $E_f$ respectively; and $\lambda$ is a weighting parameter between 0 and 1. If not specified, $n$ is the number of subjects, and $m$ is the number of ROIs in this paper. The details of these energy terms are provided in the following sections.

### 2.2.1 Anatomical constraint energy

Anatomical constraint energy $E_a$ is defined to ensure that the optimized ROIs have similar anatomical locations in the atlas space (Fig.4 shows an example of ROIs of 15 randomly selected subjects in the atlas space). We model the locations for all ROIs in the atlas space using a Gaussian model (mean: $M_{X_j}$, and standard deviation: $\sigma_{X_j}$ for ROI $j$). The model parameters were estimated using the initial locations obtained from Section 2.1. Let $X_{ij}$ be the center coordinate of region $S_{ij}$ in the atlas space, then $E_a$ is expressed as

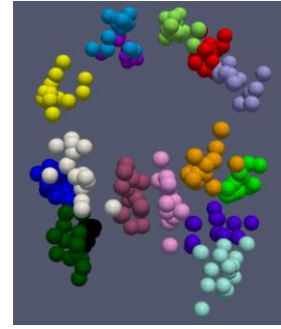

Fig.4. ROI distributions in Atlas space.

$$E_a = \begin{cases} 1 & (dmax \le 1) \\ e^{dmax-1} & (dmax > 1) \end{cases} \qquad (2)$$

where

$$dmax = Max \left\{ \left\| \frac{X_{ij} - M_{X_j}}{3\sigma_{X_j}} \right\|, 1 \le i \le n; \ 1 \le j \le m. \right\} \qquad (3)$$

Under the above definition, if any $X_{ij}$ is within the range of $3s_{X_j}$ from the distribution model center $M_{X_j}$, the anatomical constraint energy will always be one; if not, there will be an exponential increase of the energy which punishes the possible involvement of outliers. In other words, this energy factor will ensure the optimized ROIs will not significantly deviate away from the original ROIs.

### 2.2.2 Structural connectivity constraint energy

Structural connectivity constraint energy $E_c$ is defined to ensure the group has similar structural connectivity profiles for each functional ROI, since similar functional regions should have the similar structural connectivity patterns [19],

$$E_c = \sum_{i=1}^{n} \sum_{j=1}^{m} \sqrt{(C_{ij} - M_{C_j}) Covc^{-1} (C_{ij} - M_{C_j})^T} \qquad (4)$$

where $C_{ij}$ is the connectivity pattern vector for ROI $j$ of subject $i$, $M_{C_j}$ is the group mean for ROI $j$, and $Covc^{-1}$ is the inverse of the covariance matrix.

The connectivity pattern vector $C_{ij}$ is a fiber target region distribution histogram. To obtain this histogram, we first parcellate all the cortical surfaces into nine regions (as shown in Fig.5a, four lobes for each hemisphere, and the subcortical region) using the HAMMER algorithm

[17]. A finer parcellation is available but not used due to the relatively lower parcellation accuracy, which might render the histogram too sensitive to the parcellation result. Then, we extract fibers penetrating region $S_{ij}$, and calculate the distribution of the fibers' target cortical regions. Fig.5 illustrates the ideas.

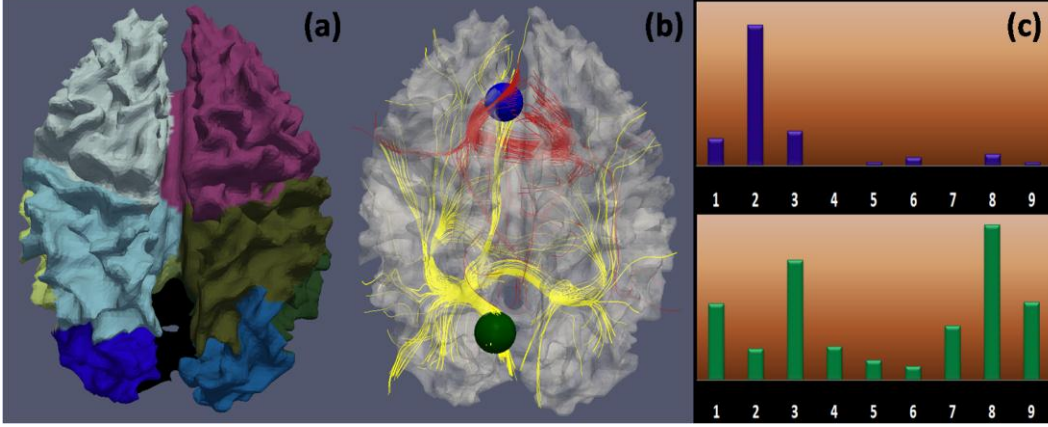

Fig.5. Structural connectivity pattern descriptor. (a): Cortical surface parcellation using HAMMER [17]; (b): Joint visualization of the cortical surface, two ROIs (blue and green spheres), and fibers penetrating the ROIs (in red and yellow, respectively); (c): Corresponding target region distribution histogram of ROIs in Fig.5b. There are nine bins corresponding to the nine cortical regions. Each bin contains the number of fibers that penetrate the ROI and are connected to the corresponding cortical region. Fiber numbers are normalized across subjects.

### 2.2.3 Functional connectivity constraint energy

Functional connectivity constraint energy $E_f$ is defined to ensure each individual has similar functional connectivity patterns for the working memory system, assuming the human brain has similar functional architecture across individuals [21].

$$E_f = \sum_{i=1}^{n} \|F_i - M_F\| \qquad (5)$$

Here, $F_i$ is the functional connectivity matrix for subject $i$, and $M_F$ is the group mean of the dataset. The connectivity between each pair of ROIs is defined using the Pearson correlation. The matrix distance used here is the Frobenius norm.

### 2.3 Energy minimization solution

The minimization of the energy defined in Section 2.2 is known as a combinatorial optimization problem. Traditional optimization methods may not fit this problem, since there are two noticeable characteristics in this application. First, we do not know how the energy changes with the varying locations of ROIs. Therefore, techniques like Newton's method cannot be used. Second, the structure of search space is not smooth, which may lead to multiple local minima during optimization. To address this problem, we adopt the simulated annealing (SA) algorithm [23] for the energy minimization. The idea of the SA algorithm is based on random walk through the space for lower energies. In these random walks, the probability of taking a step is determined by the Boltzmann distribution,

$$p = e^{-(E_{i+1} - E_i)/(KT)} \qquad (6)$$

if $E_{i+1} > E_i$, and $p = 1$ when $E_{i+1} \leq E_i$. Here, $E_i$ and $E_{i+1}$ are the system energies at solution configuration $i$ and $i+1$ respectively; $K$ is the Boltzmann constant; and $T$ is the system temperature. In other words, a step will be taken when a lower energy is found. A step will also be taken with probability $p$ if a higher energy is found. This helps avoid the local minima in the search space.

## 3 Results

Compared to structural and functional connectivity patterns, anatomical profiles are more

easily affected by variability across individuals. Therefore, the anatomical constraint energy is designed to provide constraint only to ROIs that are obviously far away from reasonableness. The reasonable range was statistically modeled by the localizations of ROIs warped into the atlas space in Section 2.2.1. Our focus in this paper is the structural and functional profiles.

### 3.1 Optimization using anatomical and structural connectivity profiles

In this section, we use only anatomical and structural connectivity profiles to optimize the locations of ROIs. The goal is to check whether the structural constraint energy $E_c$ works as expected. Fig.6 shows the fibers penetrating the right precuneus for eight subjects before (top panel) and after optimization (bottom panel). The ROI is highlighted in a red sphere for each subject. As we can see from the figure (please refer to the highlighted yellow arrows), after optimization, the third and sixth subjects have significantly improved consistency with the rest of the group than before optimization, which proves the validity of the energy function Eq.(4).

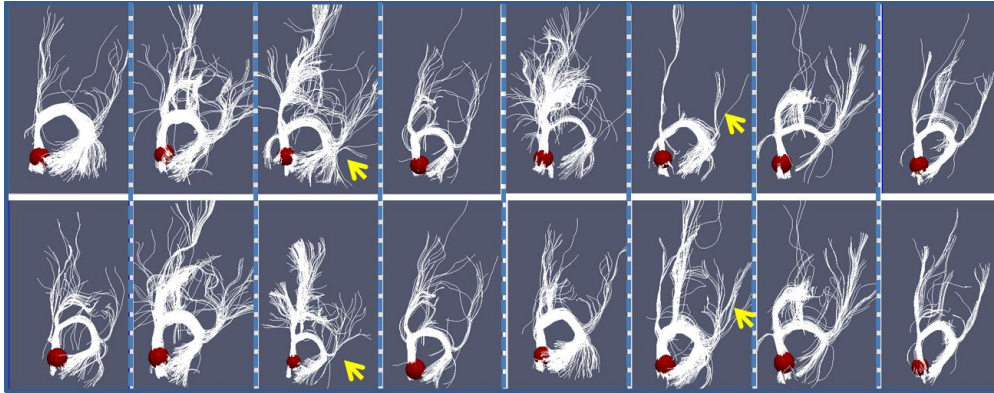

Fig.6. Comparison of structural profiles before and after optimization. Each column shows the corresponding before-optimization (top) and after-optimization (bottom) fibers of one subject. The ROI (right precuneus) is presented by the red sphere.

### 3.2 Optimization using anatomical and functional connectivity profiles

In this section, we optimize the locations of ROIs using anatomical and functional profiles, aiming to validate the definition of functional connectivity constraint energy $E_f$. If this energy constraint worked well, the functional connectivity variance of the working memory system across subjects would decrease. Fig.7 shows the comparison of the standard derivation for functional connectivity before (left) and after (right) optimization. As we can see, the variance is significantly reduced after optimization. This demonstrated the effectiveness of the defined functional connectivity constraint energy.

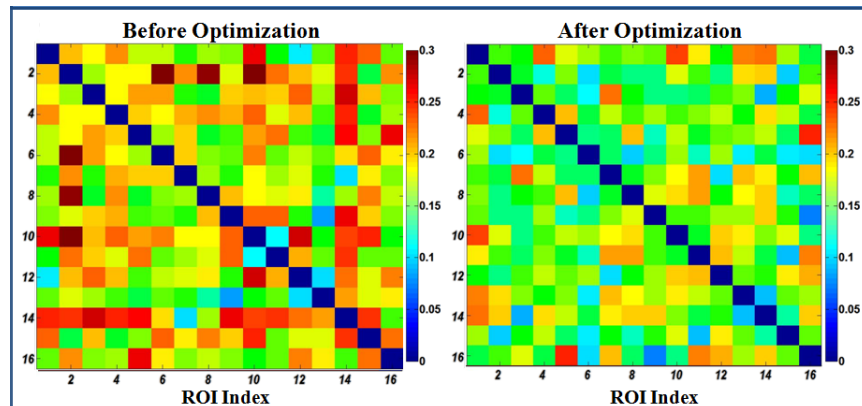

Fig.7. Comparison of the standard derivation for functional connectivity before and after the optimization. Lower values mean more consistent connectivity pattern cross subjects.

### 3.3  Consistency between optimization of functional profiles and structural profiles

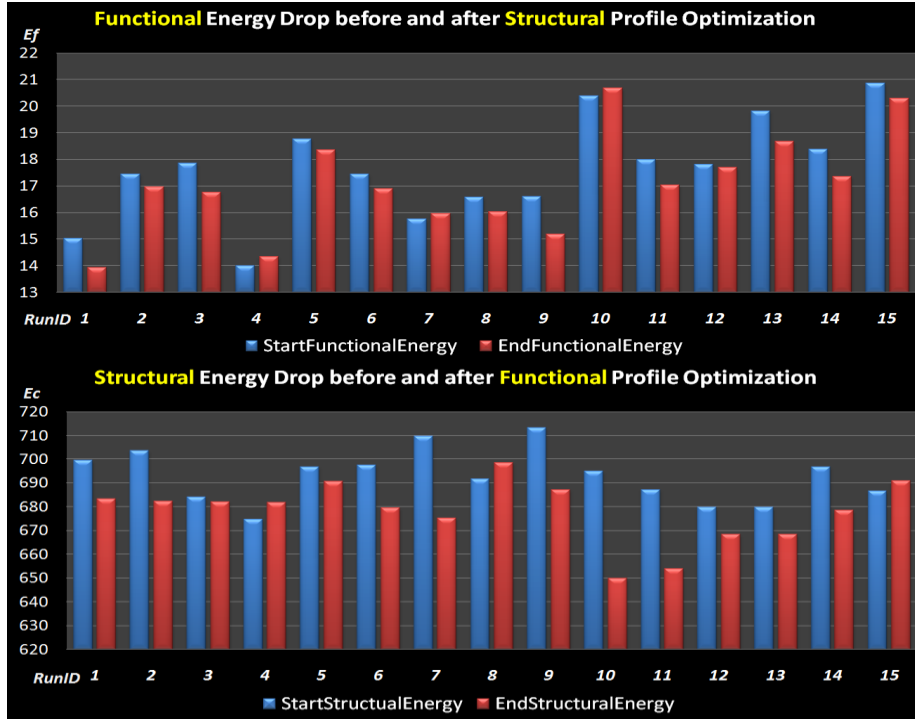

Fig.8. Optimization consistency between functional and structural profiles. Top: Functional profile energy drop along with structural profile optimization; Bottom: Structural profile energy drop along with functional profile optimization. Each experiment was repeated 15 times with random initial ROI locations that met the anatomical constraint.

The relationship between structure and function has been extensively studied [24], and it is widely believed that they are closely related. In this section, we study the relationship between functional profiles and structural profiles by looking at how the energy for one of them changes while the energy of the other decreases. The optimization processes in Section 3.1 and 3.2 were repeated 15 times respectively with random initial ROI locations that met the anatomical constraint. As shown in Fig.8, in general, the functional profile energies and structural profile energies are closely related in such a way that the functional profile energies tend to decrease along with the structural profile optimization process, while the structural profile energies also tend to decrease as the functional profile is optimized. This positively correlated decrease of functional profile energy and structural profile energy not only proves the close relationship between functional and structural profiles, but also demonstrates the consistency between functional and structural optimization, laying down the foundation of the joint optimization, whose results are detailed in the following section.

### 3.4  Optimization using anatomical, structural and functional connectivity profiles

In this section, we used all the constraints in Eq. (1) to optimize the individual locations of all ROIs in the working memory system. Ten runs of the optimization were performed using random initial ROI locations that met the anatomical constraint. Weighting parameter $\lambda$ equaled 0.5 for all these runs. Starting and ending temperatures for the simulated annealing algorithm are 8 and 0.05; Boltzmann constant $K = 1$. As we can see from Fig.9, most runs started to converge at step 24, and the convergence energy is quite close for all runs. This indicates that the simulated annealing algorithm provides a valid solution to our problem.

By visual inspection, most of the ROIs move to more reasonable and consistent locations after the joint optimization.  As an example, Fig.10 depicts the location movements of the ROI in Fig. 6 for eight subjects.  As we can see, the ROIs for these subjects share a similar anatomical

landmark, which appears to be the tip of the upper bank of the parieto-occipital sulcus. If the initial ROI was not at this landmark, it moved to the landmark after the optimization, which was the case for subjects 1, 4 and 7. The structural profiles of these ROIs are very similar to Fig.6. The results in Fig. 10 indicate the significant improvement of ROI locations achieved by the joint optimization procedure.

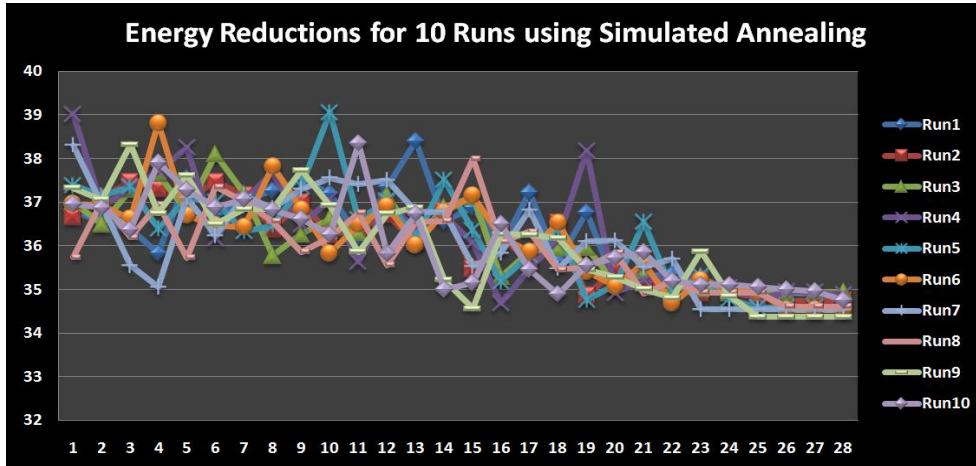

Fig.9. Convergence performance of the simulated annealing . Each run has 28 temperature conditions.

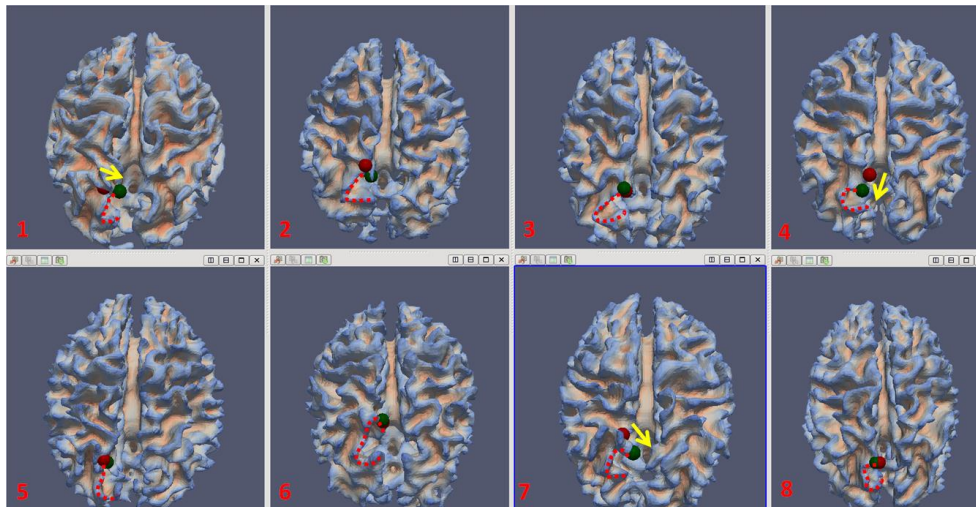

Fig.10. The movement of right precuneus before (in red sphere) and after (in green sphere) optimization for eight subjects. The "C"-shaped red dash curve for each subject depicts a similar anatomical landmark across these subjects. The yellow arrows in subject 1, 4 and 7 visualized the movement direction after optimization.

## 4      Conclusion

This paper presented a novel computational approach to optimize the locations of ROIs identified from task-based fMRI. The group-wise consistency of functional and structural connectivity patterns, and anatomical locations are jointly modeled and formulated in an energy function, which is minimized by the simulated annealing optimization algorithm. Experimental results demonstrate the optimized ROIs have more reasonable localizations, and have significantly improved the consistency of structural and functional connectivity profiles and morphological and anatomic profiles across subjects. Our future work includes extending this framework to optimize other parameters of ROIs such as size and shape, and applying and evaluating this methodology to the optimization of ROIs identified in other brain systems such as the visual, auditory, language, attention, and emotion networks.

# 5 References

1. Friston, K., Modalities, modes, and models in functional neuroimaging. Science, vol.326, no.5951, 399-403(2009).
2. Bharat B. Biswal, Toward discovery science of human brain function, PNAS March 9, 2010 vol. 107 no. 10 4734-4739.
3. Sporns O, Tononi G, Kötter R, The human connectome: A structural description of the human brain. PLoS Comput Biol. 2005 Sep; 1(4): e42.
4. Van Dijk KR, Hedden T, Venkataraman A, Evans KC, Lazar SW, Buckner RL, Intrinsic functional connectivity as a tool for human connectomics: theory, properties, and optimization. J Neurophysiol. 2010 Jan; 103(1): 297-321.
5. Hagmann P, et al., MR connectomics: Principles and challenges. J Neurosci Methods. 2010 Jan 22.
6. Friston K, J. et al., Dynamic causal modeling, Neuroimage, 19, 1273-1302, 2003.
7. http://www.humanconnectomeproject.org/
8. http://www.nitrc.org/projects/fcon_1000/
9. Goebel,R., et al., Investigating directed cortical interactions in time-resolved fMRI data using vector autoregressive modeling and Granger causality mapping. Magnetic Resonance Imaging, Volume 21, Issue 10, December 2003, Pages 1251-1261
10. Harrison L, et al., Multivariate autoregressive modeling of fMRI time series, NeuroImage, Volume 19, Issue 4, August 2003, Pages 1477-1491
11. Zang, Y., et al., "Regional homogeneity approach to fMRI data analysis," NeuroImage, 22(1): p. 394-400, 2004.
12. Jenkinson, M., Bannister, P., Brady, M., Smith, S., 2002. Improved optimization for the robust and accurate linear registration and motion correction of brain images. Neuroimage 17, 825–841.
13. Friedman, L., and Glover, G.H. (2006). Report on a Multicenter fMRI Quality Assurance Protocol. Journal of Magnetic Resonance in Imaging, 23(6):827-839.
14. H.J. Jo, J.M. Lee, J.H. Kim, C.H. Choi, B.M. Gu and D.H. Kang et al., Artificial shifting of fMRI activation localized by volume- and surface-based analyses, NeuroImage 40 (3) (2008), pp. 1077–1089.
15. W. Ou, W.M. Wells III, and P. Golland. Combining Spatial Priors and Anatomical Information for fMRI Detection. Medical Image Analysis, 14(3):318-331, 2010.
16. Tianming Liu, Hai Li, Kelvin Wong, Ashley Tarokh, Lei Guo, Stephen Wong, Brain Tissue Segmentation Based on DTI Data, NeuroImage, 38(1):114-23, 2007.
17. Shen, D., et al., 2002. HAMMER: hierarchical attribute matching mechanism for elastic registration. IEEE Trans Med Imaging 21(11), 1421-39.
18. Li K, et al., Cortical surface based identification of brain networks using high spatial resolution resting state fMRI data, International Symposium of Biomedical Imaging (ISBI) 2010.DOI: 10.1109/ISBI.2010.5490089 .
19. Passingham RE, et al., The anatomical basis of functional localization in the cortex. Nat Rev Neurosci. 3(8):606-16. 2002.
20. Van Essen, D.: A tension-based theory of morphogenesis and compact wiring in the central nervous system. Nature 385:313-318 (1997).
21. M.D. Fox and M.E. Raichle, "Spontaneous fluctuations in brain activity observed with functional magnetic resonance imaging", Nat Rev Neurosci 8:700-711, 2007.
22. Van Dijk KR, Hedden T, Venkataraman A, Evans KC, Lazar SW, Buckner RL, Intrinsic functional connectivity as a tool for human connectomics: theory, properties, and optimization. J Neurophysiol. 2010 Jan; 103(1): 297-321.
23. V. Granville, et al., Simulated annealing: A proof of convergence". IEEE Transactions on PAMI 16 (6): 652–656. June 1994.
24. Honey CJ, Sporns O, Cammoun L, Gigandet X, Thiran JP, Meuli R, Hagmann P. Predicting human resting-state functional connectivity from structural connectivity. PNAS, 106(6):2035-40. 2009.
